# Convex Clustering with Exemplar-Based Models

**Danial Lashkari**          **Polina Golland**

Computer Science and Artificial Intelligence Laboratory
Massachusetts Institute of Technology
Cambridge, MA 02139
{danial, polina}@csail.mit.edu

## Abstract

Clustering is often formulated as the maximum likelihood estimation of a mixture model that explains the data. The EM algorithm widely used to solve the resulting optimization problem is inherently a gradient-descent method and is sensitive to initialization. The resulting solution is a local optimum in the neighborhood of the initial guess. This sensitivity to initialization presents a significant challenge in clustering large data sets into many clusters. In this paper, we present a different approach to approximate mixture fitting for clustering. We introduce an exemplar-based likelihood function that approximates the exact likelihood. This formulation leads to a convex minimization problem and an efficient algorithm with *guaranteed convergence to the globally optimal solution*. The resulting clustering can be thought of as a probabilistic mapping of the data points to the set of exemplars that minimizes the average distance and the information-theoretic cost of mapping. We present experimental results illustrating the performance of our algorithm and its comparison with the conventional approach to mixture model clustering.

## 1   Introduction

Clustering is one of the most basic problems of unsupervised learning with applications in a wide variety of fields. The input is either *vectorial* data, that is, vectors of data points in the feature space, or *proximity* data, the pairwise similarity or dissimilarity values between the data points. The choice of the clustering cost function and the optimization algorithm employed to solve the problem determines the resulting clustering [1]. Intuitively, most methods seek *compact* clusters of data points, namely, clusters with relatively small intra-cluster and high inter-cluster distances. Other approaches, such as Spectral Clustering [2], look for clusters of more complex shapes lying on some low dimensional manifolds in the feature space. These methods typically transform the data such that the manifold structures get mapped to compact point clouds in a different space. Hence, they do not remove the need for efficient compact-cluster-finding techniques such as $k$-means.

The widely used Soft $k$-means method is an instance of maximum likelihood fitting of a mixture model through the EM algorithm. Although this approach yields satisfactory results for problems with a small number of clusters and is relatively fast, its use of a gradient-descent algorithm for minimization of a cost function with many local optima makes it sensitive to initialization. As the search space grows, that is, the number of data points or clusters increases, it becomes harder to find a good initialization. This problem often arises in emerging applications of clustering for large biological data sets such as gene-expression. Typically, one runs the algorithm many times with different random initializations and selects the best solution. More sophisticated initialization methods have been proposed to improve the results but the challenge of finding good initialization for EM algorithm remains [4].

We aim to circumvent the initialization procedure by designing a convex problem whose *global* optimum can be found with a simple algorithm. It has been shown that mixture modeling can

be formulated as an instance of iterative distance minimization between two sets of probability distributions [3]. This formulation shows that the non-convexity of mixture modeling cost function comes from the parametrization of the model components . More precisely, any mixture model is, by definition, a convex combination of some set of distributions. However, for a fixed number of mixture components, the set of all such mixture models is usually not convex when the distributions have, say, free mean parameters in the case of normal distributions. Inspired by combinatorial, non-parametric methods such as $k$-medoids [5] and affinity propagation [6], our main idea is to employ the notion of *exemplar* finding, namely, finding the data points which could best describe the data set. We assume that the clusters are dense enough such that there is always a data point very close to the real cluster centroid and, thus, restrict the set of possible cluster means to the set of data points. Further, by taking all data points as exemplar candidates, the modeling cost function becomes convex. A variant of EM algorithm finds the globally optimal solution.

Convexity of the cost function means that the algorithm will *unconditionally* converge to the global minimum. Moreover, since the number of clusters is not specified a priori, the algorithm *automatically finds the number of clusters* depending only on one temperature-like parameter. This parameter, which is equivalent to a common fixed variance in case of Gaussian models, defines the width scale of the desired clusters in the feature space. Our method works exactly in the same way with both proximity and vectorial data, unifying their treatment and providing insights into the modeling assumptions underlying the conversion of feature vectors into pairwise proximity data.

In the next section, we introduce our maximum likelihood function and the algorithm that maximizes it. In Section 3, we make a connection to the Rate-Distortion theory as a way to build intuition about our objective function. Section 4 presents implementation details of our algorithm. Experimental results comparing our method with a similar mixture model fitting method are presented in Section 5, followed by a discussion of the algorithm and the related work in Section 6.

## 2   Convex Cost Function

Given a set of data points $\mathcal{X} = \{\mathbf{x}_1, \cdots, \mathbf{x}_n\} \subset I\!\!R^d$, mixture model clustering seeks to maximize the scaled log-likelihood function

$$l(\{q_j\}_{j=1}^k, \{\mathbf{m}_j\}_{j=1}^k; \mathcal{X}) = \frac{1}{n} \sum_{i=1}^n \log \left[ \sum_{j=1}^k q_j f(\mathbf{x}_i; \mathbf{m}_j) \right], \qquad (1)$$

where $f(\mathbf{x}; \mathbf{m})$ is an exponential family distribution on random variable $\mathbf{X}$. It has been shown that there is a bijection between regular exponential families and a broad family of divergences called *Bregman divergence* [7]. Most of the well-known distance measures, such as Euclidean distance or Kullback-Leibler divergence (KL-divergence) are included in this family. We employ this relationship and let our model be an exponential family distribution on $\mathbf{X}$ of the form $f(\mathbf{x}; \mathbf{m}) = C(\mathbf{x}) \exp(-d_\phi(\mathbf{x}, \mathbf{m}))$ where $d_\phi$ is some Bregman divergence and $C(\mathbf{x})$ is independent of $\mathbf{m}$. Note that with this representation, $\mathbf{m}$ is the expected value of $\mathbf{X}$ under the distribution $f(\mathbf{x}; \mathbf{m})$. For instance, taking Euclidean distance as the divergence, we obtain normal distribution as our model $f$.

In this work, we take models of the above form whose parameters $\mathbf{m}$ lie in the same space as data vectors. Thus, we can restrict the set of mixture components to the distributions centered at the data points, i.e., $\mathbf{m}_j \in \mathcal{X}$. Yet, for a specified number of clusters $k$, the problem still has a combinatorial nature of choosing the right $k$ cluster centers among $n$ data points. To avoid this problem, we increase the number of possible components to $n$ and represent all data points as cluster-center candidates. The new log-likelihood function is

$$l(\{q_j\}_{j=1}^n; \mathcal{X}) = \frac{1}{n} \sum_{i=1}^n \log \sum_{j=1}^n q_j f_j(\mathbf{x}_i) = \frac{1}{n} \sum_{i=1}^n \log \left[ \sum_{j=1}^n q_j e^{-\beta d_\phi(\mathbf{x}_i, \mathbf{x}_j)} \right] + \text{const.}, \quad (2)$$

where $f_j(\mathbf{x})$ is an exponential family member with its expectation parameter equal to the $j$th data vector and the constant denotes a term that does not depend on the unknown variables $\{q_j\}_{j=1}^n$. The constant scaling factor $\beta$ in the exponent controls the sharpness of mixture components. We maximize $l(\cdot; \mathcal{X})$ over the set of all mixture distributions $\mathcal{Q} = \left\{ Q | Q(\cdot) = \sum_{j=1}^n q_j f_j(\cdot) \right\}$.

The log-likelihood function (2) can be expressed in terms of the KL-divergence by defining $\hat{P}(\mathbf{x}) = 1/n, \mathbf{x} \in \mathcal{X}$, to be the empirical distribution of the data on $I\!R^d$ and by noting that

$$D(\hat{P}\|Q) = -\sum_{\mathbf{x}\in\mathcal{X}} \hat{P}(\mathbf{x})\log Q(\mathbf{x}) - \mathbb{H}(\hat{P}) = -l(\{q_j\}_{j=1}^n; \mathcal{X}) + \text{const.} \tag{3}$$

where $\mathbb{H}(\hat{P})$ is the entropy of the empirical distribution and does not depend on the unknown mixture coefficients $\{q_j\}_{j=1}^n$. Consequently, the maximum likelihood problem can be equivalently stated as the minimization of the KL-divergence between $\hat{P}$ and the set of mixture distributions $\mathcal{Q}$.

It is easy to see that unlike the unconstrained set of mixture densities considered by the likelihood function (1), set $\mathcal{Q}$ is convex. Our formulation therefore leads to a convex minimization problem. Furthermore, it is proved in [3] that for such a problem, the sequence of distributions $Q^{(t)}$ with corresponding weights $\{q_j^{(t)}\}_{j=1}^n$ defined iteratively via

$$q_j^{(t+1)} = q_j^{(t)} \sum_{\mathbf{x}\in\mathcal{X}} \frac{\hat{P}(\mathbf{x})f_j(\mathbf{x})}{\sum_{j'=1}^n q_{j'}^{(t)} f_{j'}(\mathbf{x})} \tag{4}$$

is guaranteed to converge to the global optimum solution $Q^*$ if the support of the initial distribution is the entire index set, i.e., $q_j^{(0)} > 0$ for all $j$.

## 3 Connection to Rate-Distortion Problems

Now, we present an equivalent statement of our problem on the product set of exemplars and data points. This alternative formulation views our method as an instance of lossy data compression and directly implies the optimality of the algorithm (4).

The following proposition is introduced and proved in [3]:

**Proposition 1.** Let $\mathcal{Q}'$ be the set of distributions of the complete data random variable $(J, \mathbf{X}) \in \{1, \cdots, n\} \times I\!R^d$ with elements $Q'(j, \mathbf{x}) = q_j f_j(\mathbf{x})$. Let $\mathcal{P}'$ be the set of all distributions on the same random variable $(J, \mathbf{X})$ which have $\hat{P}$ as their marginal on $\mathbf{X}$. Then,

$$\min_{Q\in\mathcal{Q}} D(\hat{P}\|Q) = \min_{P'\in\mathcal{P}', Q'\in\mathcal{Q}'} D(P'\|Q') \tag{5}$$

where $\mathcal{Q}$ is the set of all marginal distributions of elements of $\mathcal{Q}'$ on $\mathbf{X}$. Furthermore, if $Q^*$ and $(P'^*, Q'^*)$ are the corresponding optimal arguments, $Q^*$ is the marginal of $Q'^*$.

This proposition implies that we can express our problem of minimizing (3) as minimization of $D(P'\|Q')$ where $P'$ and $Q'$ are distributions of the random variable $(J, \mathbf{X})$. Specifically, we define:

$$Q'(j, \mathbf{x}) = q_j C(\mathbf{x}) e^{-\beta d_\phi(\mathbf{x}, \mathbf{x}_j)} \qquad P'(j, \mathbf{x}) = \hat{P}(\mathbf{x})P'(j|\mathbf{x}) = \begin{cases} \frac{1}{n} r_{ij}, & \mathbf{x} = \mathbf{x}_i \in \mathcal{X}; \\ 0, & \text{otherwise} \end{cases} \tag{6}$$

where $q_j$ and $r_{ij} = P'(j|\mathbf{x} = \mathbf{x}_i)$ are probability distributions over the set $\{j\}_{j=1}^n$. This formulation ensures that $P' \in \mathcal{P}'$, $Q' \in \mathcal{Q}'$ and the objective function is expressed only in terms of variables $q_j$ and $P'(j|\mathbf{x})$ for $\mathbf{x} \in \mathcal{X}$. Our goal is then to solve the minimization problem in the space of distributions of random variable $(J, I) \in \{j\}_{j=1}^n \times \{j\}_{j=1}^n$, namely, in the product space of exemplar $\times$ data point indices. Substituting expressions (6) into the KL-divergence $D(P'\|Q')$, we obtain the equivalent cost function:

$$D(P'\|Q') = \frac{1}{n} \sum_{i,j=1}^n r_{ij}\left[\log\frac{r_{ij}}{q_j} + \beta d_\phi(\mathbf{x}_i, \mathbf{x}_j)\right] + \text{const.} \tag{7}$$

It is straightforward to show that for any set of values $r_{ij}$, setting $q_j = \frac{1}{n}\sum_i r_{ij}$ minimizes (7). Substituting this expression into the cost function, we obtain the final expression

$$\begin{aligned} D(P'\|Q'^*(P')) &= \frac{1}{n} \sum_{i,j=1}^n r_{ij}\left[\log\frac{r_{ij}}{\frac{1}{n}\sum_{i'} r_{i'j}} + \beta d_\phi(\mathbf{x}_i, \mathbf{x}_j)\right] + \text{const.} , \\ &= \mathbb{I}(I; J) + \beta\mathbb{E}_{I,J} d_\phi(\mathbf{x}_i, \mathbf{x}_j) + \text{const.} \end{aligned} \tag{8}$$

where the first term is the mutual information between the random variables $I$ (data points) and $J$ (exemplars) under the distribution $P'$ and the second term is the expected value of the pairwise distances with the same distribution on indices. The $n^2$ unknown values of $r_{ij}$ lie on $n$ separate $n$-dimensional simplices. These parameters have the same role as cluster *responsibilities* in soft $k$-means: they stand for the probability of data point $\mathbf{x}_i$ choosing data point $\mathbf{x}_j$ as its cluster-center. The algorithm described in (4) is in fact the same as the standard Arimoto-Blahut algorithm [10] commonly used for solving problems of the form (8).

We established that the problem of maximizing log-likelihood function (2) is equivalent to the minimization of objective function (8). This helps us to interpret this problem in the framework of Rate-Distortion theory. The data set can be thought of as an information source with a uniform distribution on the alphabet $\mathcal{X}$. Such a source has entropy $\log n$, which means that any scheme for encoding an infinitely long i.i.d. sequence generated by this source requires on average this number of bits per symbol, i.e., has a rate of at least $\log n$. We cannot compress the information source beyond this rate without tolerating some distortion, when the original data points are encoded into other points with nonzero distances between them. We can then consider $r_{ij}$'s as a probabilistic encoding of our data set onto itself with the corresponding average distortion $D = \mathbb{E}_{I,J} d_\phi(\mathbf{x}_i, \mathbf{x}_j)$ and the rate $\mathbb{I}(I; J)$. A solution $r_{ij}^*$ that minimizes (8) for some $\beta$ yields the least rate that can be achieved having no more than the corresponding average distortion $D$. This rate is usually denoted by $R(D)$, a function of average distortion, and is called the rate-distortion function [8]. Note that we have $\partial R/\partial D = -\beta, 0 < \beta < \infty$ at any point on the rate-distortion function graph. The weight $q_j$ for the data point $\mathbf{x}_j$ is a measure of how likely this point is to appear in the compressed representation of the data set, i.e., to be an exemplar. Here, we can rigorously quantify our intuitive idea that higher number of clusters (corresponding to higher rates) is the inherent cost of attaining lower average distortion. We will see an instance of this rate-distortion trade-off in Section 5.

## 4 Implementation

The implementation of our algorithm costs two matrix-vector multiplications per iteration, that is, has a complexity of order $n^2$ per iteration, if solved with no approximations. Letting $s_{ij} = \exp(-\beta d_\phi(\mathbf{x}_i, \mathbf{x}_j))$ and using two auxiliary vectors $z$ and $\eta$, we obtain the simple update rules

$$z_i^{(t)} = \sum_{j=1}^{n} s_{ij} q_j^{(t)} \qquad \eta_j^{(t)} = \frac{1}{n} \sum_{i=1}^{n} \frac{s_{ij}}{z_i^{(t)}} \qquad q_j^{(t+1)} = \eta_j^{(t)} q_j^{(t)} \qquad (9)$$

where the initialization $q_j^{(0)}$ is nonzero for all the data points we want to consider as possible exemplars. At the fixed point, the values of $\eta_j$ are equal to 1 for all data points in the support of $q_j$ and are less than 1 otherwise [10]. In practice, we compute the gap between $\max_j \left( \log \eta_j \right)$ and $\sum_j q_j \log \eta_j$ in each iteration and stop the algorithm when this gap becomes less than a small threshold. Note that the soft assignments $r_{ij}^{(t)} = q_j^{(t)} s_{ij}/n z_i^{(t)}$ need to be computed only once after the algorithm has converged.

Any value of $\beta \in [0, \infty)$ yields a different solution to (8) with different number of nonzero $q_j$ values. Smaller values of $\beta$ correspond to having wider clusters and greater values correspond to narrower clusters. Neither extreme, one assigning all data points to the central exemplar and the other taking all data points as exemplars, is interesting. For reasonable ranges of $\beta$, the solution is sparse and the resulting number of nonzero components of $q_j$ determines the final number of clusters.

Similar to other interior-point methods, the convergence of our algorithm becomes slow as we move close to the vertices of the probability simplex where some $q_j$'s are very small. In order to improve the convergence rate, after each iteration, we identify all $q_j$'s that are below a certain threshold ($10^{-3}/n$ in our experiments,) set them to zero and re-normalize the entire distribution over the remaining indices. This effectively excludes the corresponding points as possible exemplars and reduces the cost of the following iterations.

In order to further speed up the algorithm for very large data sets, we can search over values of $s_{ij}$ for any $i$ and keep only the largest $n_o$ values in any row turning the proximity matrix into a sparse one. The reasoning is simply that we expect any point to be represented in the final solution with exemplars relatively close to it. We observed that as long as $n_o$ values are a few times greater than the expected number of data points in each cluster, the final results remain almost the same

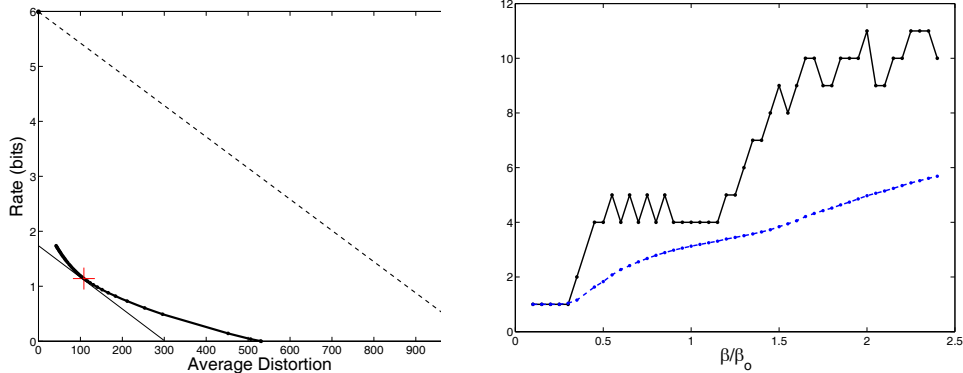

Figure 1: Left: rate-distortion function for the example described in the text. The line with slope $-\beta_o$ is also illustrated for comparison (dotted line) as well as the point corresponding to $\beta = \beta_o$ (cross) and the line tangent to the graph at that point. Right: the exponential of rate (dotted line) and number of hard clusters for different values of beta (solid line.) The rate is bounded above by logarithm of number of clusters.

with or without this preprocessing. However, this approximation decreases the running time of the algorithm by a factor $n/n_o$.

## 5  Experimental Results

To illustrate some general properties of our method, we apply it to the set of $400$ random data points in $\mathbb{R}^2$ shown in Figure 2. We use Euclidean distance and run the algorithm for different values of $\beta$. Figure 1 (left) shows the resulting rate-distortion function for this example. As we expect, the estimated rate-distortion function is smooth, monotonically decreasing and convex. To visualize the clustering results, we turn the soft responsibilities into hard assignments. Here, we first choose the set of exemplars to be the set of all indices $j$ that are MAP estimate exemplars for some data point $i$ under $P'(j|\mathbf{x}_i)$. Then, any point is assigned to its closest exemplar. Figure 2 illustrates the shapes of the resulting hard clusters for different values of $\beta$. Since $\beta$ has dimensions of inverse variance in the case of Gaussian models, we chose an empirical value $\beta_o = n^2 \log n / \sum_{i,j} \|\mathbf{x}_i - \mathbf{x}_j\|^2$ so that values $\beta$ around $\beta_o$ give reasonable results. We can see how clusters split when we increase $\beta$. Such cluster splitting behavior also occurs in the case of a Gaussian mixture model with unconstrained cluster centers and has been studied as the phase transitions of a corresponding statistical system [9]. The nature of this connection remains to be further investigated.

The resulting number of hard clusters for different values of $\beta$ are shown in Figure 1 (right). The figure indicates two regions of $\beta$ with relatively stable number of clusters, namely 4 and 10, while other cluster numbers have a more transitory nature with varying $\beta$. The distribution of data points in Figure 2 shows that this is a reasonable choice of number of clusters for this data set. However, we also observe some fluctuations in the number of clusters even in the more stable regime of values of $\beta$. Comparing this behavior with the monotonicity of our rate shows how, by turning the soft assignments into the hard ones, we lose the strong optimality guarantees we have for the original soft solution. Nevertheless, since our *global optimum* is minimum to a well justified cost function, we expect to obtain relatively good hard assignments. We further discuss this aspect of the formulation in Section 6.

The main motivation for developing a convex formulation of clustering is to avoid the well-known problem of local optima and sensitivity to initialization. We compare our method with a regular mixture model of the form (1) where $f(\mathbf{x}; \mathbf{m})$ is a Gaussian distribution and the problem is solved using the EM algorithm. We will refer to this regular mixture model as the soft $k$-means. The $k$-means algorithm is a limiting case of this mixture-model problem when $\beta \rightarrow \infty$, hence the name soft $k$-means. The comparison will illustrate how employing convexity helps us better explore the search space as the problem grows in complexity. We use synthetic data sets by drawing points from unit variance Gaussian distributions centered around a set of vectors.

There is an important distinction between the soft $k$-means and our algorithm: although the results of both algorithms depend on the choice of $\beta$, only the soft $k$-means needs the number of clusters $k$ as an input. We run the two algorithms for five different values of $\beta$ which were empirically found

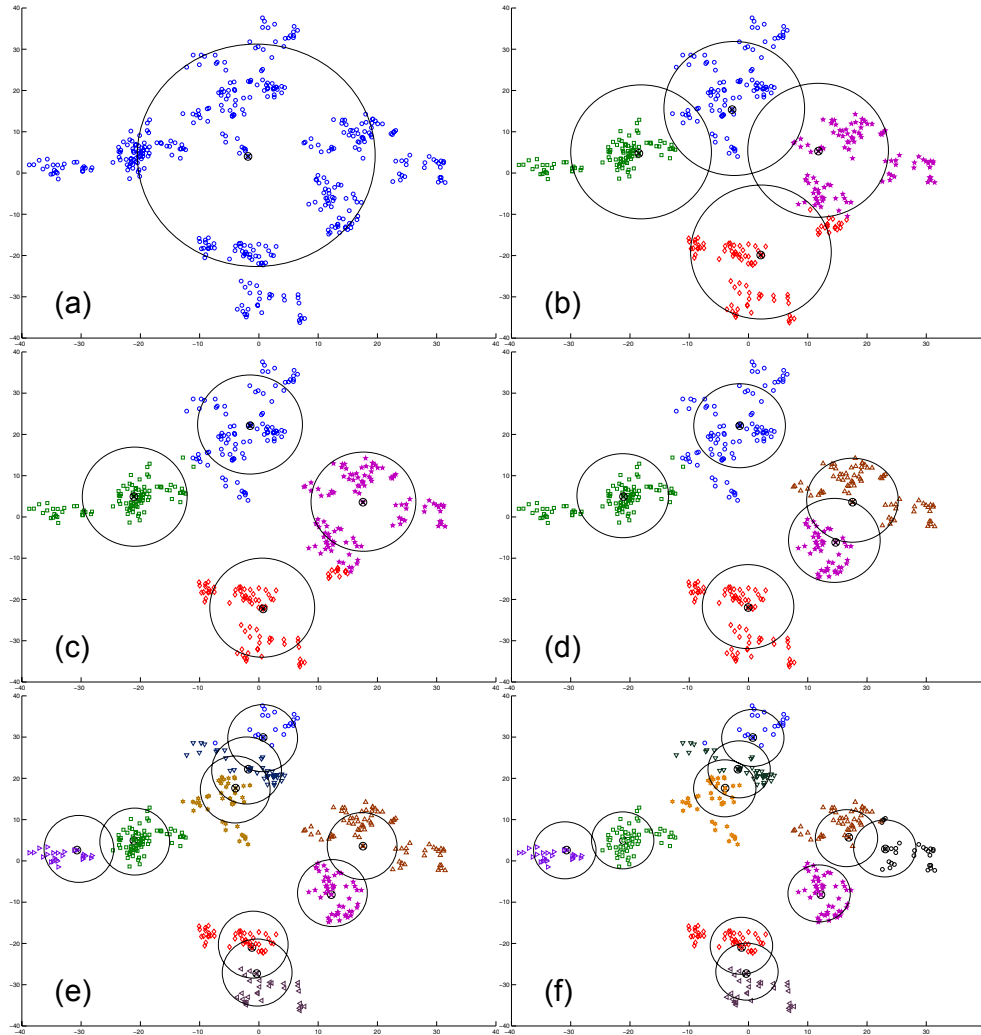

Figure 2: The clusters found for different values of $\beta$, (a) $0.1\beta_o$ (b) $0.5\beta_o$ (c) $\beta_o$ (d) $1.2\beta_o$ (e) $1.6\beta_o$ (f) $1.7\beta_o$. The exemplar data point of each cluster is denoted by a cross. The range of normal distributions for any mixture model is illustrated here by circles around these exemplar points with radius equal to the square root of the variance corresponding to the value of $\beta$ used by the algorithm ($\sigma = (2\beta)^{-1/2}$). Shapes and colors denote cluster labels.

to yield reasonable results for the problems presented here. As a measure of clustering quality, we use *micro-averaged precision*. We form the contingency tables for the cluster assignments found by the algorithm and the true cluster labels. The percentage of the total number of data points assigned to the right cluster is taken as the precision value of the clustering result. Out of the five runs with different values of $\beta$, we take the result with the best precision value for any of the two algorithms.

In the first experiment, we look at the performance of the two algorithms as the number of clusters increases. Different data sets are generated by drawing 3000 data points around some number of cluster centers in $\mathbb{R}^{20}$ with all clusters having the same number of data points. Each component of any data-point vector comes from an independent Gaussian distribution with unit variance around the value of the corresponding component of its cluster center. Further, we randomly generate components of the cluster-center vectors from a Gaussian distribution with variance 25 around zero.

In this experiment, for any value of $\beta$, we repeat soft $k$-means 1000 times with random initialization and pick the solution with the highest likelihood value. Figure 3 (left) presents the precision values as a function of the number of clusters in the mixture distribution that generates the 3000 data points. The error bars summarize the standard deviation of precision over 200 independently generated data sets. We can see that performance of soft $k$-means drops as the number of clusters increases while our performance remains relatively stable. Consequently, as illustrated in Figure 3 (right),

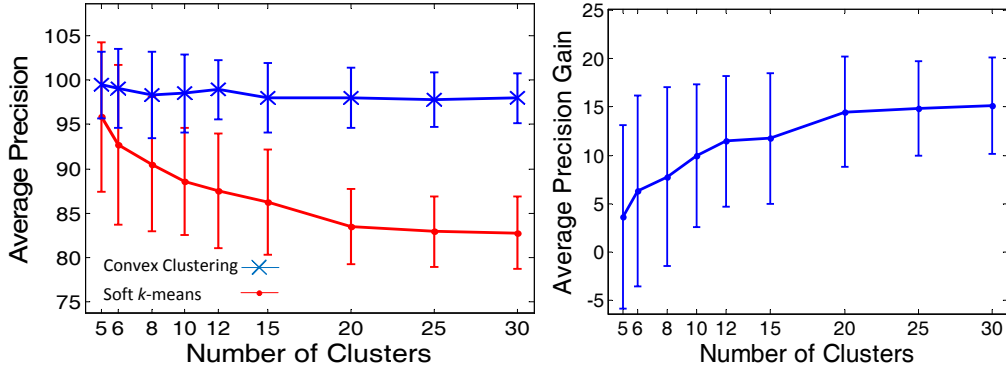

Figure 3: Left: average precision values of Convex Clustering and Soft $k$-means for different numbers of clusters in 200 data sets of 3000 data points. Right: precision gain of using Convex Clustering in the same experiment.

the average precision difference of the two algorithms increases with increasing number of clusters. Since the total number of data points remains the same, increasing the number of clusters results in increasing complexity of the problem with presumably more local minima to the cost function. This trend agrees with our expectation that the results of the convex algorithm improves relative to the original one with a larger search space.

As another way of exploring the complexity of the problem, in our second experiment, we generate data sets with different dimensionality. We draw 100 random vectors, with unit variance Gaussian distribution in each component, around any of the 40 cluster centers to make data sets of total 4000 data points. The cluster centers are chosen to be of the form $(0, \cdots, 0, \sqrt{50}, 0, \cdots, 0)$ where we change the position of the nonzero component to make different cluster centers. In this way, the pairwise distance between all cluster centers is 50 by formation.

Figure 4 (left) presents the precision values found for the two algorithms when 4000 points lie in spaces with different dimensionality. Soft $k$-means was repeated 100 times with random initialization for any value of $\beta$. Again, the relative performance of Convex Clustering when compared to soft $k$-means improves with the increasing problem complexity. This is another evidence that for larger data sets the less precise nature of our constrained search, as compared to the full mixture models, is well compensated by its ability to always find its global optimum. In general the value of $\beta$ should be tuned to find the desired solution. We plan to develop a more systematic way for choosing $\beta$.

## 6 Discussion and Related Work

Since only the distances take part in our formulation and the values of data point vectors are not required, we can extend this method to any proximity data. Given a matrix $D_{n \times n} = [d_{ij}]$ that describes the pairwise symmetric or asymmetric dissimilarities between data points, we can replace $d_\phi(\mathbf{x}_i, \mathbf{x}_j)$'s in (8) with $d_{ij}$'s and solve the same minimization problem whose convexity can be directly verified. The algorithm works in exactly the same way and all the aforementioned properties carry over to this case as well.

A previous application of rate-distortion theoretic ideas in clustering led to the deterministic annealing (DA). In order to avoid local optima, DA gradually decreases an annealing parameter, tightening the bound on the average distortion [9]. However, at each temperature the same standard EM updates are used. Consequently, the method does not provide strong guarantees on the global optimality of the resulting solution.

Affinity propagation is another recent exemplar-based clustering algorithm. It finds the exemplars by forming a factor graph and running a message passing algorithm on the graph as a way to minimize the clustering cost function [6]. If the data point $i$ is represented by the data point $c_i$, assuming a common preference parameter value $\lambda$ for all data points, the objective function of affinity propagation can be stated as $\sum_i d_{ic_i} + \lambda k$ where $k$ is the number of found clusters. The second term is needed to put some cost on picking any point as an exemplar to prevent the trivial case of sending any point to itself. Outstanding results have been reported for the affinity propagation [6] but theoretical guarantees on its convergence or optimality are yet to be established.

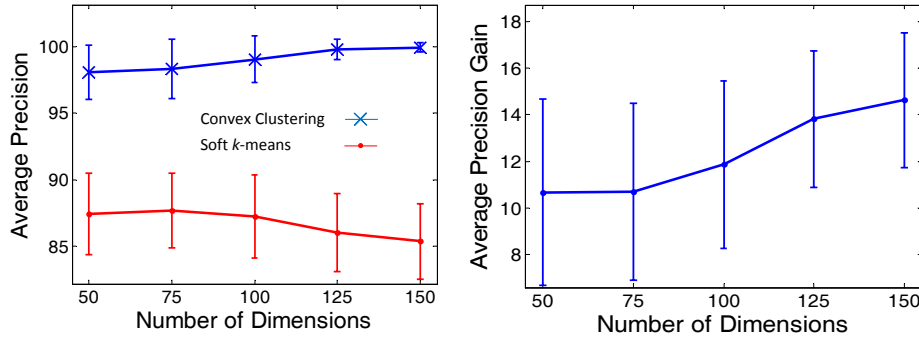

Figure 4: Left: average precision values of Convex Clustering and Soft $k$-means for different data dimensionality in 100 data sets of 4000 data points with 40 clusters. Right: precision gain of using Convex Clustering in the same experiment.

We can interpret our algorithm as a relaxation of this combinatorial problem to the soft assignment case by introducing probabilities $\mathbb{P}(c_i = j) = r_{ij}$ of associating point $i$ with an exemplar $j$. The marginal distribution $q_j = \frac{1}{n} \sum_i r_{ij}$ is the probability that point $j$ is an exemplar. In order to use analytical tools for solving this problem, we have to turn the regularization term $k$ into a continuous function of assignments. A possible choice might be $\mathbb{H}(q)$, entropy of distribution $q_j$, which is bounded above by $\log k$. However, the entropy function is concave and any local or global minimum of a concave minimization problem over a simplex occurs in an extreme point of the feasible domain which in our case corresponds to the original combinatorial hard assignments [11]. In contrast, using mutual information $\mathbb{I}(I, J)$ induced by $r_{ij}$ as the regularizing term turns the problem into a convex problem. Mutual information is convex and serves as a lower bound on $\mathbb{H}(q)$ since it is always less than the entropy of both of its random variables. Now, by letting $\lambda = 1/\beta$ we arrive to our cost function in (8). We can therefore see that our formulation is a convex relaxation of the original combinatorial problem.

In conclusion, we proposed a framework for constraining the search space of general mixture models to achieve global optimality of the solution. In particular, our method promises to be useful in problems with large data sets where regular mixture models fail to yield consistent results due to their sensitivity to initialization. We also plan to further investigate generalization of this idea to the models with more elaborate parameterizations.

**Acknowledgements**. This research was supported in part by the NIH NIBIB NAMIC U54-EB005149, NCRR NAC P41-RR13218 grants and by the NSF CAREER grant 0642971.

## References

[1] J. Puzicha, T. Hofmann, and J. M. Buhmann, "Theory of proximity based clustering: Structure detection by optimization," *Pattern Recognition*, Vol. 33, No. 4, pp. 617–634, 2000.

[2] A. Y. Ng, M. I. Jordan, and Y. Weiss, "On Spectral Clustering: Analysis and an Algorithml," *Advances in Neural Information Processing Systems*, Vol. 14, pp. 849–856, 2001.

[3] I. Csiszár and P. Shields, "Information Theory and Statistics: A Tutorial," *Foundations and Trends in Communications and Information Theory*, Vol. 1, No. 4, pp. 417–528, 2004.

[4] M. Meilă, and D. Heckerman, "An Experimental Comparison of Model-Based Clustering Methods," *Machine Learning*, Vol. 42, No. 1-2, pp. 9–29, 2001.

[5] J. Han, and M. Kamber, *Data Mining: Concepts and Techniques*, Morgan Kaufmann, 2001.

[6] B. J. Frey, and D. Dueck, "Clustering by Passing Messages Between Data Points," *Science*, Vol. 315, No. 5814, pp. 972–976, 2007.

[7] A. Banerjee, S. Merugu, I. S.Dhillon, and J. Ghosh, "Clustering with Bregman Divergences," *Journal of Machine Learning Research*, Vol. 6, No. 6, pp. 1705-1749, 2005.

[8] T. M. Cover, and J. A. Thomas, *Elements of information theory*, New York, Wiley, 1991.

[9] K. Rose, "Deterministic Annealing for Clustering, Compression, Classification, Regression, and Related Optimization Problems," *Proceedings of the IEEE*, Vol. 86, No. 11, pp. 2210–2239, 1998.

[10] R. E. .Blahut, "Computation of Channel Capacity and Rate-Distortion Functions," *IEEE Transactions on Information Theory*, Vol. IT-18, No. 4, pp. 460–473, 1974.

[11] M. Pardalos, and J. B. Rosen, "Methods for Global Concave Minimization: A Bibliographic Survey," *SIAM Review*, Vol. 28, No. 3., pp. 367–379, 1986.

